# Learning spike-based correlations and conditional probabilities in silicon

**Aaron P. Shon    David Hsu    Chris Diorio**
Department of Computer Science and Engineering
University of Washington
Seattle, WA 98195-2350 USA
*{aaron, hsud, diorio}@cs.washington.edu*

## Abstract

We have designed and fabricated a VLSI synapse that can learn a conditional probability or correlation between spike-based inputs and feedback signals. The synapse is low power, compact, provides nonvolatile weight storage, and can perform simultaneous multiplication and adaptation. We can calibrate arrays of synapses to ensure uniform adaptation characteristics. Finally, adaptation in our synapse does not necessarily depend on the signals used for computation. Consequently, our synapse can implement learning rules that correlate past and present synaptic activity. We provide analysis and experimental chip results demonstrating the operation in learning and calibration mode, and show how to use our synapse to implement various learning rules in silicon.

## 1   Introduction

Computation with conditional probabilities and correlations underlies many models of neurally inspired information processing. For example, in the sequence-learning neural network models proposed by Levy[1], synapses store the log conditional probability that a presynaptic spike occurred given that the postsynaptic neuron spiked sometime later. Boltzmann machine synapses learn the difference between the correlations of pairs of neurons in the sleep and wake phase[2]. In most neural models, computation and adaptation occurs at the synaptic level. Hence, a silicon synapse that can learn conditional probabilities or correlations between pre- and post-synaptic signals can be a key part of many silicon neural-learning architectures.

We have designed and implemented a silicon synapse, in a 0.35 $\mu$m CMOS process, that learns a synaptic weight that corresponds to the conditional probability or correlation between binary input and feedback signals. This circuit utilizes floating-gate transistors to provide both nonvolatile storage and weight adaptation mechanisms[3]. In addition, the circuit is compact, low power, and provides simultaneous adaptation and computation. Our circuit improves upon previous implementations of floating-gate based learning synapses[3,4,5] in several ways.

First, our synapse appears to be the first spike-based floating-gate synapse that implements a general learning principle, rather than a particular learning rule[4,5]. We demon-

strate that our synapse can learn either the conditional probability or the correlation between input and feedback signals. Consequently, we can implement a wide range of synaptic learning networks with our circuit.

Second, unlike the general correlational learning synapse proposed by Hasler et. al. [3], our synapse can implement learning rules that correlate pre-and postsynaptic activity that occur at different times. Learning algorithms that employ time-separated correlations include both temporal difference learning [6] and recently postulated temporally asymmetric Hebbian learning [7]. Hasler's correlational floating-gate synapse can only perform updates based on the present input and feedback signals, and is therefore unsuitable for learning rules that correlate signals that occur at different times. Because signals that control adaptation and computation in our synapse are separate, our circuit can implement these time-dependent learning rules.

Finally, we can calibrate our synapses to remove mismatch between the adaptation mechanisms of individual synapses. Mismatch between the same adaptation mechanisms on different floating-gate transistors limits the accuracy of learning rules based on these devices. This problem has been noted in previous circuits that use floating-gate adaptation [4,8]. In our circuit, different synapses can learn widely divergent weights from the same inputs because of component mismatch. We provide a calibration mechanism that enables identical adaptation across multiple synapses despite device mismatch. To our knowledge, this circuit is the first instance of a floating-gate learning circuit that includes this feature.

This paper is organized as follows. First, we provide a brief introduction to floating-gate transistors. Next, we provide a description and analysis of our synapse, demonstrating that it can learn the conditional probability or correlation between a pair of binary signals. We then describe the calibration circuitry and show its effectiveness in compensating for adaptation mismatches. Finally, we discuss how this synapse can be used for silicon implementations of various learning networks.

## 2  Floating-gate transistors

Because our circuit relies on floating-gate transistors to achieve adaptation, we begin by briefly discussing these devices. A floating-gate transistor (e.g. transistor $M_3$ of Fig. 1(a)) comprises a MOSFET whose gate is isolated on all sides by $SiO_2$. A control gate capacitively couples signals to the floating gate. Charge stored on the floating gate implements a nonvolatile analog weight; the transistor's output current varies with both the floating-gate voltage and the control-gate voltage. We use Fowler-Nordheim tunneling [9] to increase the floating-gate charge, and impact-ionized hot-electron injection (IHEI) [10] to decrease the floating-gate charge. We tunnel by placing a high voltage on a tunneling implant, denoted by the arrow in Fig. 1(a). We inject by imposing more than about 3V across the drain and source of transistor $M_3$. The circuit allows simultaneous adaptation and computation, because neither tunneling nor IHEI interfere with circuit operation. Over a wide range of tunneling voltages $V_{tun}$, we can approximate the magnitude of the tunneling current $I_{tun}$ as [4]:

$$I_{tun} = I_{tun0} \exp\left(V_{tun} - V_{fg}\right)/V_\chi \tag{1}$$

where $V_{tun}$ is the tunneling-implant voltage, $V_{fg}$ is the floating-gate voltage, and $I_{tun0}$ and $V_\chi$ are fit constants. Over a wide range of transistor drain and source voltages, we can approximate the magnitude of the injection current $I_{inj}$ as [4]:

$$I_{inj} = I_{inj0} I_s^{1-U_t/V_\gamma} \exp\left((V_s - V_d)/V_\gamma\right) \tag{2}$$

where $V_s$ and $V_d$ are the drain and source voltages, $I_{inj0}$ is a pre-exponential current, $V_\gamma$ is a constant that depends on the VLSI process, and $U_t$ is the thermal voltage kT/q.

# 3 The silicon synapse

We show our silicon synapse in Fig. 1. The synapse stores an analog weight $W$, multiplies $W$ by a binary input $X_{in}$, and adapts $W$ to either a conditional probability P($X_{cor}|Y$) or a correlation P($X_{cor}Y$). $X_{in}$ is analogous to a presynaptic input, while $Y$ is analogous to a postsynaptic signal or error feedback. $X_{cor}$ is a presynaptic adaptation signal, and typically has some relationship with $X_{in}$. We can implement different learning rules by altering the relationship between $X_{cor}$ and $X_{in}$. For some examples, see section 4.

We now describe the circuit in more detail. The drain current of floating-gate transistor $M_4$ represents the weight value $W$. Because the control gate of M$_4$ is fixed, $W$ depends solely on the charge on floating-gate capacitor C$_1$. We can switch the drain current on or off using transistor M$_7$; this switching action corresponds to a multiplication of the weight value $W$ by a binary input signal, $X_{in}$. We choose values for the drain voltage of the M$_4$ to prevent injection. A second floating-gate transistor M$_3$, whose gate is also connected to C$_1$, controls adaptation by injection and tunneling. Simultaneously high input signals $X_{cor}$ and $Y$ cause injection, increasing the weight. A high $V_{tun}$ causes tunneling, decreasing the weight. We either choose to correlate a high $V_{tun}$ with signal Y or provide a fixed high $V_{tun}$ throughout the adaptation process. The choice determines whether the circuit learns a conditional probability or a correlation, respectively.

Because the drain current sourced by M$_4$ provides is the weight $W$, we can express $W$ in terms of M$_4$'s floating-gate voltage, $V_{fg}$. $V_{fg}$ includes the effects of both the fixed control-gate voltage and the variable floating-gate charge. The expression differs depending on whether the readout transistor is operating in the subthreshold or above-threshold regime. We provide both expressions below:

$$W = \begin{cases} I_0 \exp(-\kappa^2 V_{fg}/(1+\kappa)U_t) & \text{below threshold} \\ \beta\left(V_0 - \dfrac{\kappa^2 V_{fg}}{(1+\kappa)}\right)^2 & \text{above threshold} \end{cases} \tag{3}$$

Here $V_0$ is a constant that depends on the threshold voltage and on $V_{dd}$, $U_t$ is the thermal voltage kT/q, $\kappa$ is the floating-gate-to-channel coupling coefficient, and I$_0$ is a fixed bias current. Eq. 3 shows that $W$ depends solely on $V_{fg}$, (all the other factors are constants). These equations differ slightly from standard equations for the source current through a transistor due to source degeneration caused by M$_4$. This degeneration smoothes the nonlinear relationship between $V_{fg}$ and $I_s$; its addition to the circuit is optional.

## 3.1 Weight adaptation

Because $W$ depends on $V_{fg}$, we can control $W$ by tunneling or injecting transistor M$_3$. In this section, we show that these mechanisms enable our circuit to learn the correlation or conditional probability between inputs $X_{cor}$ (which we will refer to as $X$) and $Y$. Our analysis assumes that these statistics are fixed over some period during which adaptation occurs. The change in floating-gate voltage, and hence the weight, discussed below should therefore be interpreted in terms of the expected weight change due to the statistics of the inputs. We discuss learning of conditional probabilities; a slight change in the tunneling signal, described previously, allows us to learn correlations instead.

We first derive the injection equation for the floating-gate voltage in terms of the joint probability P($X,Y$) by considering the relationship between the input signals and $I_s$, $V_s$,

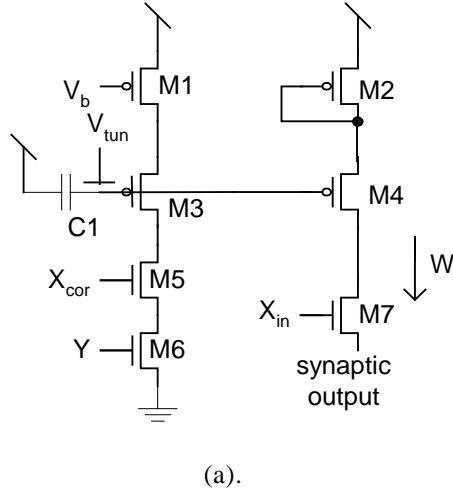

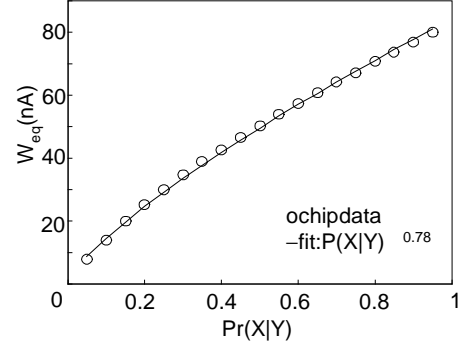

(b)

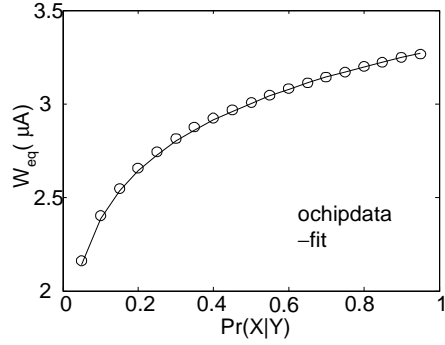

(a).

Fig. 1. (a) Synapse schematic. (b) Plot of equilibrium weight in the subthreshold regime versus the conditional probability P($X|Y$), showing both experimental chip data and a fit from Eq. 7 (c). Plot of equilibrium weight versus conditional probability in the above-threshold regime, again showing chip data and a fit from Eq. 7.

(c)

and $V_d$ of M$_3$. We assume that transistor M$_1$ is in saturation, constraining $I_s$ at M$_3$ to be constant. Presentation of a joint binary event ($X,Y$) closes nFET switches M$_5$ and M$_6$, pulling the drain voltage $V_d$ of M$_3$ to 0V and causing injection. Therefore the probability that $V_d$ is low enough to cause injection is the probability of the joint event Pr($X,Y$). By Eq. 2, the amount of the injection is also dependent on M$_3$'s source voltage $V_s$. Because M$_3$ is constrained to a fixed channel current, a drop in the floating-gate voltage, $\Delta V_{fg}$, causes a drop in $V_s$ of magnitude $\kappa\Delta V_{fg}$. Substituting these expressions into Eq. 2 results in a floating-gate voltage update of:

$$(dV_{fg}/dt)_{inj} = -I_{inj0}\Pr(X,Y)\exp(\kappa V_{fg}/V_\chi) \tag{4}$$

where $I_{inj0}$ also includes the constant source current. Eq. 4 shows that the floating-gate voltage update due to injection is a function of the probability of the joint event ($X,Y$).

Next we analyze the effects of tunneling on the floating-gate voltage. The origin of the tunneling signal determines whether the synapse is learning a conditional probability or a correlation. If the circuit is learning a conditional probability, occurrence of the conditioning event $Y$ gates a corresponding high-voltage (~9V) signal onto the tunneling implant. Consequently, we can express the change in floating-gate voltage due to tunneling in terms of the probability of $Y$, and the floating-gate voltage.

$$(dV_{fg}/dt)_{tun} = I_{tun0}\Pr(Y)\exp(-V_{fg}/V_\chi) \tag{5}$$

Eq. 5 shows that the floating-gate voltage update due to tunneling is a function of the probability of the event $Y$.

### 3.2 Weight equilibrium

To demonstrate that our circuit learns $P(X|Y)$, we show that the equilibrium weight of the synapse is solely a function of $P(X|Y)$. The equilibrium weight of the synapse is the weight value where the expected weight change over time equals zero. This weight value corresponds to the floating-gate voltage where injection and tunneling currents are equal. To find this voltage, we equate Eq.'s 4 and 5 and solve:

$$V_{fg}^{eq} = \frac{-1}{\left(\kappa/V_y + 1/V_x\right)}\left(\log \Pr(X \mid Y) + \log \frac{I_{inj0}}{I_{tun0}}\right) \qquad (6)$$

To derive the equilibrium weight, we substitute Eq. 6 into Eq. 3 and solve:

$$W_{eq} = \begin{cases} I_0\left(\dfrac{I_{inj0}}{I_{tun0}}\Pr(X \mid Y)\right)^{\alpha} & \text{below threshold} \\[2ex] \beta\left(V_0 + \eta\left(\log \dfrac{I_{inj0}}{I_{tun0}} + \log\left(\Pr(X \mid Y)\right)\right)\right)^2 & \text{above threshold} \end{cases} \qquad (7)$$

$$\text{where } \alpha = \frac{\kappa^2}{(1+\kappa)U_t(\kappa/V_\gamma + 1/V_\chi)} \text{ and } \eta = \frac{\kappa^2}{(1+\kappa)(\kappa/V_\gamma + 1/V_\chi)}.$$

Consequently, the equilibrium weight is a function of the conditional probability below threshold and a function of the log-squared conditional probability above threshold. Note that the equilibrium weight is stable because of negative feedback in the tunneling and injection processes. Therefore, the weight will always converge to the equilibrium value shown in Eq. 7. Figs. 1(b) and (c) show the equilibrium weight versus the conditional $P(X|Y)$ for both sub- and above-threshold circuits, along with fits to Eq. 7.

Note that both the sub- and above-threshold relationship between $P(X|Y)$ and the equilibrium weight enables us to compute the probability of a vector of synaptic inputs $X$ given a post-synaptic response $Y$. In both cases, we can apply the outputs currents of an array of synapses through diodes, and then add the resulting voltages via a capacitive voltage divider, resulting in a voltage that is a linear function of log $P(X|Y)$.

### 3.3 Calibration circuitry

Mismatch between injection and tunneling in different floating-gate transistors can greatly reduce the ability of our synapses to learn meaningful values. Experimental data from floating-gate transistors fabricated in a 0.35 μm process show that injection varies by as much as 2:1 across a chip, and tunneling by up to 1.2:1. The effect of this mismatch on our synapses causes the weight equilibrium of different synapses to differ by a multiplicative gain. Fig. 2(b) shows the equilibrium weights of an array of six synapses exposed to identical input signals. The variation of the synaptic weights is of the same order of magnitude as the weights themselves, making large arrays of synapses all but useless for implementing many learning algorithms.

We alleviate this problem by calibrating our synapses to equalize the pre-exponential tunneling and injection constants. Because the dependence of the equilibrium weight on these constants is determined by the ratio of $I_{inj0}/I_{tun0}$, our calibration process changes $I_{inj}$ to equalize the ratio of injection to tunneling across all synapses. We choose to calibrate injection because we can easily change $I_{inj0}$ by altering the drain current through M[1].

Our calibration procedure is a self-convergent memory write[11], that causes the equilibrium weight of every synapse to equal the current $I_{cal}$. Calibration requires many operat-

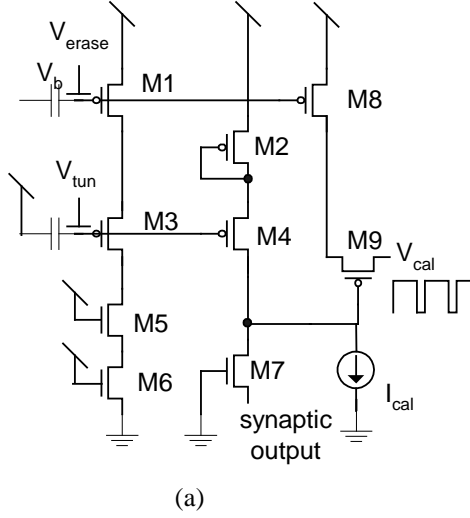

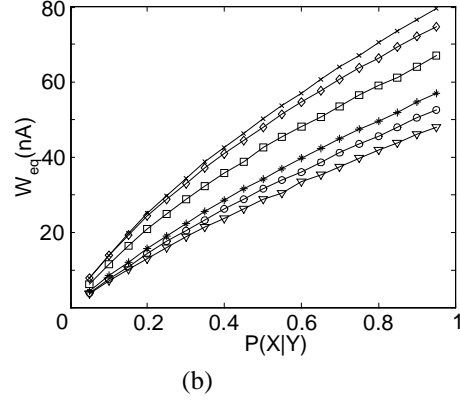

(b)

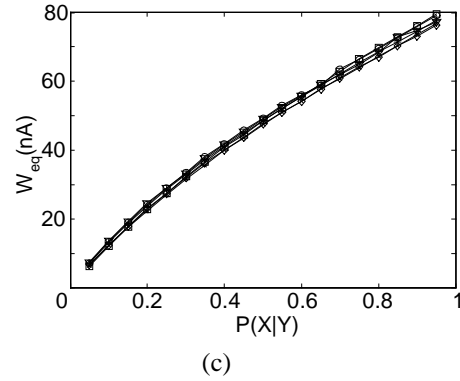

(c)

Fig. 2. (a) Schematic of calibrated synapse with signals used during the calibration procedure. (b) Equilibrium weights for array of synapses shown in Fig. 1a. (c) Equilibrium weights for array of calibrated synapses after calibration.

ing cycles, where, during each cycle, we first increase the equilibrium weight of the synapse, and second, we let the synapse adapt to the new equilibrium weight.

We create the calibrated synapse by modifying our original synapse according to Fig. 2(a). We convert $M_1$ into a floating-gate transistor, whose floating-gate charge thereby sets $M_3$'s channel current, providing control of $I_{inj0}$ of Eq. 7. Transistor $M_8$ modifies $M_1$'s gate charge by means of injection when $M_9$'s gate is low and $V_{cal}$ is low. $M_9$'s gate is only low when the equilibrium weight $W$ is less than $I_{cal}$. During calibration, injection and tunneling on $M_3$ are continuously active. We apply a pulse train to $V_{cal}$; during each pulse period, $V_{cal}$ is predominately high. When $V_{cal}$ is high, the synapse adapts towards its equilibrium weight. When $V_{cal}$ pulses low, $M_8$ injects, increasing the synapse's equilibrium weight $W$. We repeat this process until the equilibrium weight $W$ matches $I_{cal}$, causing $M_9$'s gate voltage to rise, disabling $V_{cal}$ and with it injection. To ensure that a precalibrated synapse has an equilibrium weight below $I_{cal}$, we use tunneling to erase all bias transistors prior to calibration. Fig. 2(c) shows the equilibrium weights of six synapses after calibration. The data show that calibration can reduce the effect of mismatched adaptation on the synapse's learned weight to a small fraction of the weight itself.

Because $M_1$ is a floating-gate transistor, its parasitic gate-drain capacitance causes a mild dependence between $M_1$'s drain voltage and source current. Consequently, $M_3$'s floating-gate voltage now affects its source current (through $M_1$'s drain voltage), and we can model $M_3$ as a source-degenerated pFET [3]. The new expression for the injection current in $M_3$ is:

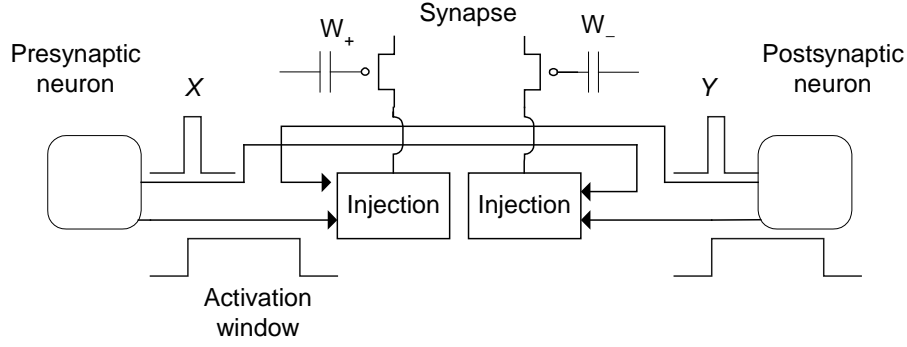

Fig. 3. A method for achieving spike-time dependent plasticity in silicon.

$$\left( \frac{dV_{fg}}{dt} \right)_{inj} = -I_{inj0} \Pr(X, Y) \exp\left( V_{fg} \left( \frac{\kappa}{V_\gamma} - \frac{\kappa k_1}{U_t} \right) \right) \tag{8}$$

where $k_1$ is close to zero. The new expression for injection slightly changes the $\alpha$ and $\eta$ terms of the weight equilibrium in Eq. 7, although the qualitative relationship between the weight equilibrium and the conditional probability remains the same.

## 4 Implementing silicon synaptic learning rules

In this section we discuss how to implement a variety of learning rules from the computational-neurobiology and neural-network literature with our synapse circuit.

We can use our circuit to implement a Hebbian learning rule. Simultaneously activating both $M_5$ and $M_6$ is analogous to heterosynaptic LTP based on synchronized pre- and post-synaptic signals, and activating tunneling with the postsynaptic $Y$ is analogous to homo-synaptic LTD. In our synapse, we tie $X_{in}$ and $X_{cor}$ together and correlate $V_{tun}$ with $Y$.

Our synapse is also capable of emulating a Boltzmann weight-update rule [2]. This weight-update rule derives from the difference between correlations among neurons when the network receives external input, and when the network operates in a free running phase (denoted as clamped and unclamped phases respectively). With weight decay, a Boltzmann synapse learns the difference between correlations in the clamped and unclamped phase. We can create a Boltzmann synapse from a pair of our circuits, in which the effective weight is the difference between the weights of the two synapses. To implement a weight update, we update one silicon synapse based on pre- and postsynaptic signals in the clamped phase, and update the other synapse in the unclamped phase. We do this by sending $X_{in}$ to $X_{cor}$ of one synapse in the clamped phase, and sending $X_{in}$ to $X_{cor}$ of the other synapse in the negative phase. $V_{tun}$ remains constant throughout adaptation.

Finally, we consider implementing a temporally asymmetric Hebbian learning rule [7] using our synapse. In temporally asymmetric Hebbian learning, a synapse exhibits LTP or LTD if the presynaptic input occurs before or after the postsynaptic response, respectively. We implement an asymmetric learning synapse using two of our circuits, where the synaptic weight is the difference in the weights of the two circuit. We show the circuit in Fig. 3. Each neuron sends two signals: a neuronal output, and an adaptation time window that is active for some time afterwards. Therefore, the combined synapse receives two presynaptic signals and two postsynaptic signals. The relative timing of a postsynaptic response, $Y$, with the presynaptic input, $X$, determines whether the synapse undergoes

LTP or LTD. If *Y* occurs before *X*, *Y*'s time window correlates with *X*, causing injection on the negative synapse, decreasing the weight. If *Y* occurs after *X*, *Y* correlates with *X*'s time window, causing injection on the positive synapse, increasing the weight. Hence, our circuit can use the relative timing between presynaptic and postsynaptic activity to implement learning.

## 5 Conclusion

We have described a silicon synapse that implements a wide range of spike-based learning rules, and that does not suffer from device mismatch. We have also described how we can implement various silicon-learning networks using this synapse. In addition, although we have only analyzed the learning properties of the synapse for binary signals, we can instead use pulse-coded analog signals. One possible avenue for future work is to analyze the implications of different pulse-coded schemes on the circuit's adaptive behavior.

### Acknowledgements

This work was supported by the National Science Foundation and by the Office of Naval Research. Aaron Shon was also supported by a NDSEG fellowship. We thank Anhai Doan and the anonymous reviewers for helpful comments.

## References

[1]  W. B. Levy, "A computational approach to hippocampal function," in R. D. Hawkins and G. H. Bower (eds.), *Computational Models of Learning in Simple Neural Systems*, *The Psychology of Learning and Motivation* vol. 23, pp. 243-305, San Diego, CA: Academic Press, 1989.

[2]  D. H. Ackley, G. Hinton, and T. Sejnowski, "A learning algorithm for Boltzmann machines," *Cognitive Science* vol. 9, pp. 147-169, 1985.

[3]  P. Hasler, B. A. Minch, J. Dugger, and C. Diorio, "Adaptive circuits and synapses using pFET floating-gate devices, " in G. Cauwenberghs and M. Bayoumi (eds.) *Learning in Silicon*, pp. 33-65, Kluwer Academic, 1999.

[4]  P. Hafliger, *A spike-based learning rule and its implementation in analog hardware*, Ph.D. thesis, ETH Zurich, 1999.

[5]  C. Diorio, P. Hasler, B. A. Minch, and C. Mead, "A floating-gate MOS learning array with locally computer weight updates," *IEEE Transactions on Electron Devices* vol. 44(12), pp. 2281-2289, 1997.

[6]  R. Sutton, "Learning to predict by the methods of temporal difference," *Machine Learning,* vol. 3, pp. 9-44, 1988.

[7]  H. Markram, J. Lübke, M. Frotscher, and B. Sakmann, "Regulation of synaptic efficacy by coincidence of postsynaptic APs and EPSPs," *Science* vol. 275, pp. 213-215, 1997.

[8]  A. Pesavento, T. Horiuchi, C. Diorio, and C. Koch, " Adaptation of current signals with floating-gate circuits," in *Proceedings of the 7[th] International Conference on Microelectronics for Neural, Fuzzy, and Bio-Inspired Systems (Microneuro99)*, pp. 128-134, 1999.

[9]  M. Lenzlinger and E. H. Snow. "Fowler-Nordheim tunneling into thermally grown $SiO_2$," *Journal of Applied Physics* vol. 40(1), pp. 278--283, 1969.

[10] E. Takeda, C. Yang, and A. Miura-Hamada, *Hot Carrier Effects in MOS Devices*, San Diego, CA: Academic Press, 1995.

[11] C. Diorio, "A p-channel MOS synapse transistor with self-convergent memory writes," *IEEE Journal of Solid-State Circuits* vol. 36(5), pp. 816-822, 2001.
